# Adapting Codes and Embeddings for Polychotomies

**Gunnar Rätsch, Alexander J. Smola**
RSISE, CSL, Machine Learning Group
The Australian National University
Canberra, 0200 ACT, Australia
{Gunnar.Raetsch, Alex.Smola}@anu.edu.au

**Sebastian Mika**
Fraunhofer FIRST
Kekulestr. 7
12489 Berlin, Germany
mika@first.fhg.de

## Abstract

In this paper we consider formulations of multi-class problems based on a generalized notion of a margin and using output coding. This includes, but is not restricted to, standard multi-class SVM formulations. Differently from many previous approaches we learn the code as well as the embedding function. We illustrate how this can lead to a formulation that allows for solving a wider range of problems with for instance many classes or even "missing classes". To keep our optimization problems tractable we propose an algorithm capable of solving them using two-class classifiers, similar in spirit to Boosting.

## 1 Introduction

The theory of pattern recognition is primarily concerned with the case of binary classification, i.e. of assigning examples to one of two categories, such that the expected number of misassignments is minimal. Whilst this scenario is rather well understood, theoretically as well as empirically, it is not directly applicable to many practically relevant scenarios, the most prominent being the case of more than two possible outcomes.

Several learning techniques naturally generalize to an arbitrary number of classes, such as density estimation, or logistic regression. However, when comparing the reported performance of these systems with the de-facto standard of using two-class techniques in combination with simple, fixed output codes to solve multi-class problems, they often lack in terms of performance, ease of optimization, and/or run-time behavior.

On the other hand, many methods have been proposed to apply binary classifiers to multi-class problems, such as Error Correcting Output Codes (ECOC) [6, 1], Pairwise Coupling [9], or by simply reducing the problem of discriminating $C$ classes to $C$ "one vs. the rest" dichotomies. Unfortunately the optimality of such methods is not always clear (e.g., how to choose the code, how to combine predictions, scalability to many classes).

Finally, there are other problems similar to multi-class classification which can not be solved satisfactory by just combining simpler variants of other algorithms: multi-label problems, where each instance should be assigned to a subset of possible categories, and ranking problems, where each instance should be assigned a rank for all or a subset of possible outcomes. These problems can, in reverse order of their appearance, be understood as more and more refined variants of a multi-variate regression, i.e.

$$\text{two-class} \subseteq \text{multi-class} \subseteq \text{multi-label} \subseteq \text{ranking} \subseteq \text{multi-variate regression}$$

Which framework and which algorithm in there one ever chooses, it is usually possible to make out a single scheme common to all these: There is an encoding step in which

the input data are embedded into some "code space" and in this space there is a code book which allows to assign one or several labels or ranks respectively by measuring the similarity between mapped samples and the code book entries. However, most previous work is either focused on finding a good embedding given a fixed code or just optimizing the code, given a fixed embedding (cf. Section 2.3).

The aim of this work is to propose (i) a multi-class formulation which optimizes the code *and* the embedding of the training sample into the code space, and (ii) to develop a general ranking technique which as well specializes to specific multi-class, multi-label and ranking problems as it allows to solve more general problems. As an example of the latter consider the following model problem: In chemistry people are interested in mapping sequences to structures. It is not yet known if there is an one-to-one correspondence and hence the problem is to find for each sequence the best matching structures. However, there are only say a thousand sequences the chemists have good knowledge about. They are assigned, with a certain rank, to a subset of say a thousand different structures. One could try to cast this as a standard multi-class problem by assigning each training sequence to the structure ranked highest. But then, there will be classes to which only very few or no sequences are assigned and one can obviously hardly learn using traditional techniques. The machine we propose is (at least in principle) able to solve problems like this by reflecting relations between classes in the way the code book is constructed and at the same time trying to find an embedding of the data space into the code space that allows for a good discrimination.

The remainder of this paper is organized as follows: In Section 2 we introduce some basic notions of large margin, output coding and multi-class classification. Then we discuss the approaches of [4] and [21] and propose to learn the code book. In Section 3 we propose a rather general idea to solve resulting multi-class problems using two-class classifiers. Section 4 presents some preliminary experiments before we conclude.

## 2  Large Margin Multi-Class Classification

Denote by $\mathcal{X}$ the sample space (not necessarily a metric space), by $\mathcal{Y}$ the space of possible labels or ranks (e.g. $\mathcal{Y} = 1, \ldots, C$ for multi-class problems where $C$ denotes the number of classes, or $\mathcal{Y} = \mathbb{R}^C$ for a ranking problem), and let $Z$ be a training sample of size $N$, i.e. $Z = \{(\boldsymbol{x}_n, y_n) \in \mathcal{X} \times \mathcal{Y} \text{ with } n \in 1, \ldots, N\}$.

**Output Coding**  It is well known (see [6, 1] and references therein) that multi-class problems can be solved by decomposing a polychotomy into $L$ dichotomies and solving these separately using a two-class technique. This can be understood as assigning to each class $c$ a binary string $\boldsymbol{t}(c) \in \{-1, 1\}^L$ of length $L$ which is called a *code word*. This results in an $C \times L$ binary code matrix. Now each of the $L$ columns of this matrix defines a partitioning of $C$ classes into two subsets, forming binary problems for which a classifier is trained. Evaluation is done by computing the output of all $L$ learned functions, forming a new bit-string, and then choosing the class $c$ such that some distance measure between this string and the corresponding row of the code matrix is minimal, usually the Hamming distance. Ties can be broken by uniformly selecting a winning class, using prior information or, where possible, using confidence outputs from the basic classifiers.[1]

Since the codes for each class must be unique, there are $\binom{2^L}{C} = \mathcal{O}(2^{CL})$ (for $C \ll 2^L$) possible code matrices to choose from. One possibility is to choose the codes to be error-correcting (ECOC) [6]. Here one uses a code book with e.g. large Hamming distance between the code words, such that one still gets the correct decision even if a few of the classifiers err. However, finding the code that minimizes the training error is NP-complete, even for fixed binary classifiers [4]. Furthermore, errors committed by the binary classifiers are not necessarily independent, significantly reducing the effective number of wrong bits that one can handle [18, 19]. Nonetheless ECOC has proven useful and algorithms for finding a good code (and partly also finding the corresponding classifiers) have been

proposed in e.g. [15, 7, 1, 19, 4]. Noticeably, most practical approaches suggest to drop the requirement of binary codes, and instead propose to use continuous ones.

We now show how predictions with small (e.g. Hamming) distance to their appropriated code words can be related to a large margin classifier, beginning with binary classification.

## 2.1 Large Margins

**Dichotomies**   Here a large margin classifier is defined as a mapping $f : \mathcal{X} \to \mathbb{R}$ with the property that $yf(\boldsymbol{x})$, or more specifically $y_n f(\boldsymbol{x}_n) \geq \rho$ with $(\boldsymbol{x}_n, y_n) \in Z$, where $\rho$ is some positive constant [20]. Since such a positive margin may not always be achievable, one typically maximizes a penalized version of the maximum margin, such as

$$\frac{\lambda}{N} \sum_{n=1}^{N} \epsilon_n + \Omega[f] \text{ where } y_n f(\boldsymbol{x}_n) \geq 1 - \epsilon_n, \ \epsilon_n \geq 0, \ n = 1, \ldots, N \text{ and } f \in \mathcal{F}. \quad (1)$$

Here $\Omega[f]$ is a regularization term, $\lambda > 0$ is a regularization constant and $\mathcal{F}$ denotes the class of functions under consideration. Note that for $y_n = 1$ we could rewrite the condition $f(\boldsymbol{x}_n) \geq 1 - \epsilon_n$ also as $\frac{1}{4}(f(\boldsymbol{x}_n) - (-1))^2 - \frac{1}{4}(f(\boldsymbol{x}_n) - 1)^2 \geq 1 - \epsilon_n$ (and likewise for $y_n = -1$). In other words, we can express the margin as the difference between the distance of $f(\boldsymbol{x})$ from the target $1$ and the target $-1$.

**Polychotomies**   While this insight by itself is not particularly useful, it paves the way for an extension of the notion of the margin to multi-class problems: denote by d a distance measure and by $\boldsymbol{t}(c) \in \mathbb{R}^L$, $c = 1, \ldots, C$ ($L \in \mathbb{N}_+$ is the length of the code) target vectors corresponding to class $c$. Then we can define the margin $\rho(\boldsymbol{f}, \boldsymbol{x}, y)$ of an observation $\boldsymbol{x}$ and class $y$ with respect to $\boldsymbol{f} : X \to \mathbb{R}^L$ as

$$\rho(\boldsymbol{f}, \boldsymbol{x}, y) := \min_{c \neq y} \mathrm{d}(\boldsymbol{t}(c), \boldsymbol{f}(\boldsymbol{x})) - \mathrm{d}(\boldsymbol{t}(y), \boldsymbol{f}(\boldsymbol{x})). \quad (2)$$

This means that we measure the minimal relative difference in distance between $\boldsymbol{f}$, the correct target $\boldsymbol{t}(y)$ and any other target $\boldsymbol{t}(c)$ (cf. [4]). We obtain accordingly the following optimization problem:

$$\text{minimize } \frac{\lambda}{N} \sum_{n=1}^{N} \epsilon_n + \Omega[\boldsymbol{f}] \text{ where } \mathrm{d}(\boldsymbol{t}(c), \boldsymbol{f}(\boldsymbol{x}_n)) - \mathrm{d}(\boldsymbol{t}(y_n), \boldsymbol{f}(\boldsymbol{x}_n)) \geq 1 - \epsilon_n \quad (3)$$

for all $c \neq y_n, \epsilon_n \geq 0$ and $\boldsymbol{f} \in \mathcal{F}$. For the time being we chose 1 as a reference margin — an adaptive means of choosing the reference margin can be implemented using the $\nu$-trick, which leads to an easier to control regularization parameter [16].

## 2.2 Distance Measures

Several choices of d are possible. However, one can show that only $\mathrm{d}(\boldsymbol{t}, \boldsymbol{f}) = \|\boldsymbol{t} - \boldsymbol{f}\|_2^2$ and related functions will lead to a convex constraint on $\boldsymbol{f}$:

**Lemma 1 (Difference of Distance Measures)**   *Denote by* $\mathrm{d}(\boldsymbol{t}, \boldsymbol{f}) : \mathbb{R}^L \times \mathbb{R}^L \to \mathbb{R}$ *a symmetric distance measure. Then the only case where* $\mathrm{d}(\boldsymbol{t}, \boldsymbol{f}) - \mathrm{d}(\boldsymbol{t}', \boldsymbol{f})$ *is convex in* $\boldsymbol{f}$ *for all* $\boldsymbol{t}, \boldsymbol{t}'$ *occurs if* $\mathrm{d}(\boldsymbol{t}, \boldsymbol{f}) = \mathrm{d}_{\boldsymbol{t}}(\boldsymbol{t}) + \mathrm{d}_{\boldsymbol{f}}(\boldsymbol{f}) + \boldsymbol{t}^\top B \boldsymbol{f}$, *where* $B \in \mathbb{R}^{L \times L}$ *is symmetric.*

**Proof**   Convexity in $\boldsymbol{f}$ implies that $\partial_{\boldsymbol{f}}^2 [\mathrm{d}(\boldsymbol{t}, \boldsymbol{f}) - \mathrm{d}(\boldsymbol{t}', \boldsymbol{f})]$ is positive semidefinite. This is only possible if $\partial_{\boldsymbol{f}}^2 \mathrm{d}(\boldsymbol{t}, \boldsymbol{f})$ is a function of $\boldsymbol{f}$ only. The latter, however, implies that the only joint terms in $\boldsymbol{t}$ and $\boldsymbol{f}$ must be of linear nature in $\boldsymbol{f}$. Symmetry, on the other hand, implies that the term must be linear in $\boldsymbol{t}$, too, which proves the claim.  ∎

Lemma 1 implies that any distance functions other than the ones described above will lead to optimization problems with potentially many local minima, which is not desirable. However, for quadratic d we will get a convex optimization problem (assuming suitable $\Omega[f]$)

and then there are ways to efficiently solve (3). Finally, re-defining $\tilde{t}(c) := t(c)B$ means that it is sufficient to consider only $d(t, f) = \|t - f\|^2$. We obtain

$$d(t(c), f(x)) - d(t(y), f(x)) = \|t(c)\|^2 - \|t(y)\|^2 + 2t(c)^\top f(x) - 2t(y)^\top f(x). \quad (4)$$

Note, that if the code words have the same length, the difference of the projections of $f$ onto different code words determines the margin. We will indeed later consider a more convenient case: $d(t(c), f(x)) = t(c)^\top f(x)$, which will lead to linear constraints only and allows us to use standard optimization packages. However, there are no principal limitations about using the Euclidean distance.

If we choose $t(c)$ to be an error-correcting code, such as those in [6, 1], one will often have $L < C$. Hence, we use fewer dimensions than we have classes. This means that during optimization we are trying to find C functions $g_c(x) = d(t(c), f(x))$, $c = 1, \ldots, C$, from an L dimensional subspace. In other words, we choose the subspace and perform regularization by allowing only a smaller class of functions. By appropriately choosing the subspace one may encode prior knowledge about the problem.

## 2.3 Discussion and Relation to Previous Approaches

Note that for $t(c)_l = -1 + 2\delta_{c,l}$ we have that (4) is equal $2t(c)^\top f(x) - 2t(y)^\top f(x) = 2(f_y(x) - f_c(x))$ and hence the problem of multi-class classification reverts to the problem of solving C binary classification problems of one vs. the remaining classes. Then our approach turns out to be very similar to the idea presented in [21] (except for some additional slack-variables).

A different approach was taken in [4]. Here, the function $f$ is held fix and the code $t$ is optimized. In their approach, the code is described as a vector in a kernel feature space and one obtains in fact an optimization problem very similar to the one in [21] and (3) (again, the slack-variables are defined slightly different).

Another idea which is quite similar to ours was also presented at the conference [5]. The resulting optimization problem turns out to be convex, but with the drawback, that one can either not fully optimize the code vectors or not guarantee that they are well separated.

Since these approaches were motivated by different ideas (one optimizing the code, the other optimizing the embedding), this shows that the role of the code $t(c)$ and the embedding function $f$ is interchangeable if the function or the code, respectively, is fixed.

Our approach allows arbitrary codes for which a function $f$ is learned. This is illustrated in Figure 1. The position of the code words (="class centers") determine the function $f$. The position of the centers relative to each other may reflect relationships between the classes (e.g. classes "black" & "white" and "white" & "grey" are close).

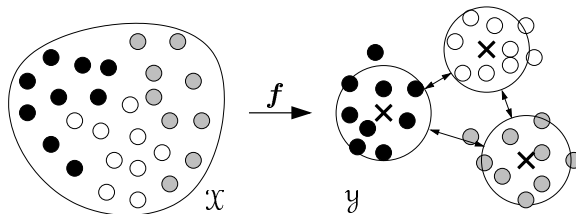

Figure 1: Illustration of embedding idea: The samples are mapped from the input space $\mathcal{X}$ into the code space $\mathcal{Y}$ via the embedding function $f$, such that samples from the same class are close to their respective code book vector (crosses on the right). The spatial organization of the code book vectors reflects the organization of classes in the $\mathcal{X}$ space.

## 2.4 Learning Code & Embedding

This leaves us with the question of how to determine a "good" code and a suitable $f$. As we can see from (4), for fixed $f$ the constraints are linear in $t$ and vice versa, yet we have non-

convex constraints, if both $\boldsymbol{f}$ and $\boldsymbol{t}$ are variable. Finding the *global optimum* is therefore computationally infeasible when optimizing $\boldsymbol{f}$ and $\boldsymbol{t}$ simultaneously (furthermore note that any rotation applied to $\boldsymbol{t}$ and $\boldsymbol{f}$ will leave the margin invariant, which shows the presence of local minima due to equivalent codes).

Instead, we propose the following method: for fixed code $\boldsymbol{t}$ optimize over $\boldsymbol{f}$, and subsequently, for fixed $\boldsymbol{f}$, optimize over $\boldsymbol{t}$, possibly repeating the process. The first step follows [4], i.e. to learn the code for a fixed function. Both steps separately can be performed fairly efficient (since the optimization problems are convex; cf. Lemma 1).

This procedure is guaranteed to decrease the over all objective function at every step and converges to a local minimum. We now show how a code maximizing the margin can be found. To avoid a trivial solution (we can may virtually increase the margin by rescaling all $\boldsymbol{t}$ by some constant), we add $\sum_{c=1}^{C} \|\boldsymbol{t}(c)\|_2^2$ to the objective function. It can be shown that one does not need an additional regularization constant in front of this term, if the distance is linear on both arguments. If one prefers sparse codes, one may use the $\ell_1$-norm instead. In summary, we obtain the following convex quadratic program for finding the codes which can be solved using standard optimization techniques:

$$
\begin{aligned}
\underset{\boldsymbol{t}(c), \epsilon_n \geq 0}{\text{minimize}} \quad & \frac{\lambda}{N} \sum_{n=1}^{N} \epsilon_n + \sum_{c=1}^{C} \|\boldsymbol{t}(c)\|_2^2 \\
\text{subject to} \quad & (\boldsymbol{t}(y_n) - \boldsymbol{t}(c))^\top \boldsymbol{f}(\boldsymbol{x}_n) \geq 1 - \epsilon_n \text{ for all } n = 1, \ldots, N \text{ and } c \neq y_n.
\end{aligned}
\tag{5}
$$

The technique for finding the embedding will be discussed in more detail in Section 3.

**Initialization** To obtain a good initial code, we may either take recourse to readily available tables [17] or we may use a random code, e.g. by generating vectors uniformly distributed on the L-dimensional sphere. One can show that the probability that there exists two such vectors (out of C) that have a smaller distance than $\epsilon$ is bounded by $\frac{1}{2}(C-1)C \arcsin^{L-1}(\epsilon/2)$ (proof given in the full paper). Hence, with probability greater than $\frac{1}{2}$ the random code vectors have distances greater than $2C^{-\frac{2}{L-1}}$ from each other.[2]

## 3 Column Generation for Finding the Embedding

There are several ways to setup and optimize the resulting optimization problem (3). For instance in [21, 4] the class of functions is the set of C hyperplanes in some kernel feature space and the regularizer $\Omega[f]$ is the sum of the $\ell_2$-norms of the hyperplane normal vectors. In this section we consider a different approach. Denote by $H := \{h_j : X \rightarrow \mathbb{R}^C | j = 1, \ldots, J\}$ a class of basis functions and let $\boldsymbol{f}_\alpha = \sum_{j=1}^{J} \alpha_j h_j \in \text{span } H$. We choose the regularizer $\Omega[f]$ to be the $\ell_1$-norm on the expansion coefficients. We are interested in solving:

$$
\begin{aligned}
\underset{\boldsymbol{\alpha} \in \mathbb{R}_+^J, \boldsymbol{\varepsilon} \in \mathbb{R}^N}{\text{minimize}} \quad & \sum_{j=1}^{J} \alpha_j + \frac{\lambda}{N} \sum_{n=1}^{N} \varepsilon_n \\
\text{subject to} \quad & (\boldsymbol{t}(c) - \boldsymbol{t}(y_n))^\top \boldsymbol{f}_\alpha(\boldsymbol{x}_n) \geq 1 - \varepsilon_n, \ n = 1, \ldots, N, \ y_n \neq c = 1, \ldots, C
\end{aligned}
\tag{6}
$$

To derive a column generation method [12, 2] we need the dual optimization problem, or more specifically its constraints: $d_{n,c} \geq 0, n = 1, \ldots, N$ and $y_n \neq c = 1, \ldots, C$,

$$
\sum_{n=1}^{N} \sum_{\tilde{y}_n \neq c = 1}^{C} d_{n,c}(\boldsymbol{t}(y_n) - \boldsymbol{t}(c))^\top h_j(\boldsymbol{x}_n) \leq 1, \ j = 1, \ldots, J
\tag{7}
$$

and $\sum_{y_n \neq c=1}^{C} d_{n,c} \leq \lambda/N, n = 1, \ldots, N$. The idea of column generation is to start with a restricted master problem, namely without the variables $\boldsymbol{\alpha}$ (i.e $J = 0$). Then one solves the corresponding dual problem (7) and then finds the hypothesis that corresponds to a violated constraint (and also one primal variable). This hypothesis is included in the optimization problem, one resolves and finds the next violated constraint. If all constraints of the full problem are satisfied, one has reached optimality.

We now construct a hypothesis set $H$ from a scalar valued base-class $\tilde{H} = \{h_{\tilde{j}} : X \rightarrow \mathbb{R}$ where $\tilde{j} = 1, \ldots, \tilde{J}\}$, which has particularly nice properties for our purposes. The idea is to extend $h_{\tilde{j}}$ by multiplication with vectors $\boldsymbol{\beta} \in \mathbb{R}^{L}$:

$$H \equiv H_p = \left\{ \boldsymbol{\beta} h : X \rightarrow \mathbb{R}^{L} \ \Big| \ \boldsymbol{\beta} \in \mathbb{R}^{L}, \|\boldsymbol{\beta}\|_p = 1, h \in \tilde{H} \right\}.$$

Since there are infinitely many functions in this set $H$, we have an infinite number of constraints in the dual optimization problem. By using the described column generation technique one can, however, find the solution of this semi-infinite programming problem [13]. We have to identify the constraint in (7), which is maximally violated, i.e. one has to find a "partitioning" $\boldsymbol{\beta}$ and a hypothesis $h_{\tilde{j}}$ with maximal

$$\sum_{n=1}^{N} \sum_{y_n \neq c=1}^{C} d_{n,c} h_{\tilde{j}}(\boldsymbol{x}_n)(\boldsymbol{t}(y_n) - \boldsymbol{t}(c))^{\top} \boldsymbol{\beta} = \boldsymbol{\beta}^{\top} A_{\tilde{j}}, \tag{8}$$

for appropriate $A_{\tilde{j}}$. Maximizing (8) with respect to $\|\boldsymbol{\beta}\|_p = 1$ is easy for a given $h_{\tilde{j}}$: for $p = \infty$, one chooses $\boldsymbol{\beta} = \operatorname{sgn}(A_{\tilde{j}})$; if $p = 2$, then $\boldsymbol{\beta} = A_{\tilde{j}}/\|A_{\tilde{j}}\|_2$ and for $p = 1$ one chooses the minimizing unit vector. However, finding $h$ and $\boldsymbol{\beta}$ simultaneously is a difficult problem, if not all $h_{\tilde{j}}$ are known in advance (see also [15]). We propose to test all previously used hypotheses to find the best $\boldsymbol{\beta}$. As a second step one finds the hypothesis $h_{\tilde{j}}$ that maximizes $\boldsymbol{\beta}^{\top} A_{\tilde{j}}$. Only if one cannot find a hypothesis that violates a constraint, one employs the more sophisticated techniques suggested in [15]. If there is no hypothesis $\boldsymbol{\beta} h$ left that corresponds to a violated constraint, the dual optimization problem is optimal.

In this work we are mainly interested in the case $p = \infty$, since then $\boldsymbol{\beta} \in \{\pm 1\}^{L}$ and the problem of finding $h_{\tilde{j}}$ simplifies greatly. Then we can use another learning algorithm that minimizes or approximately minimizes the training error of a weighted training set (rewrite (8)). This approach has indeed many similarities to Boosting. Following the ideas in [14] one can show that there is a close relationship between our technique using the trivial code and the multi-class boosting algorithms as e.g. proposed in [15].

## 4 Extensions and Illustration

### 4.1 A first Experiment

In a preliminary set of experiments we use two benchmark data sets from the UCI benchmark repository: `glass` and `iris`. We used our column generation strategy as described in Section 3 in conjunction with the code optimization problem to solve the combined optimization problem to find the code and the embedding. We used $L = C$. The algorithm has only one model parameters ($\lambda$). We selected it by 5-fold cross validation on the training data. The test error is determined by averaging over five splits of training and test data. As base learning algorithm we chose decision trees (C4.5) which we only use as two-class classifier in our column generation algorithm.

On the `glass` data set we obtained an error rate of $30.9\% \pm 5.6\%$. In [1] an error of $37\%$ was reported for SVMs using a polynomial kernel. We also computed the test error of multi-class decision trees and obtained $36.5\% \pm 5.6\%$ error. Hence, our hybrid algorithm could relatively improve existing results by $15\%$. On the `iris` data we could achieve an error rate of $4.0\% \pm 2.8\%$ and could slightly improve the result of decision trees ($6.0\% \pm 5.0\%$).

However, SVMs beat our result with $2\%$ error [1]. We conjecture that this is due to the properties of decision trees which have problems generating smooth boundaries not aligned with coordinate axes.

So far, we could only show a proof of concept and more experimental work is necessary. It is in particular interesting to find practical examples, where a non-trivial choice of the code (via optimization) helps simplifying the embedding and finally leads to additional improvements. Such problems often appear in Computer Vision, where there are strong relationships between classes. Preliminary results indicate that one can achieve considerable improvements when adapting codes and embeddings [3].

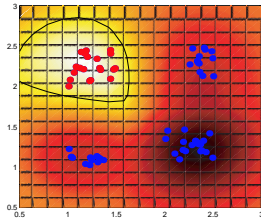

Figure 2: Toy example for learning missing classes. Shown is the decision boundary and the confidence for assigning a sample to the upper left class. The training set, however, did not contain samples from this class. Instead, we used (9) with the information that each example besides belonging to its own class with confidence two also belongs to the other classes with confidence one *iff* its distance to the respective center is less than one.

### 4.2 Beyond Multi-Class

So far we only considered the case where there is only one class to which an example belongs to. In a more general setting as for example the problem mentioned in the introduction, there can be several classes, which possibly have a ranking. We have the sets $\mathcal{R}_n \subset C = \{(c_1, c_2) \mid c_1, c_2 \in \{1, \dots, N\}\}$, which contain all pairs of "relations" between the positive classes. The set $\mathcal{S}_n \subset C$ contains all pairs of positive and negative classes of an example.

$$
\begin{aligned}
&\underset{\substack{\boldsymbol{\alpha} \in \mathbb{R}_+^J, \boldsymbol{\varepsilon} \in \mathbb{R}_+^{|\mathcal{R}|}, \\ \boldsymbol{\varepsilon}' \in \mathbb{R}_+^N, \boldsymbol{b} \in \mathbb{R}^C, \boldsymbol{t}(c_i)}}{\text{minimize}} && \sum_{j=1}^{J} \alpha_j + \lambda_{\mathcal{R}} \sum_{n=1}^{N} \sum_{r=1}^{|\mathcal{R}_n|} \varepsilon_{n,r} + \lambda_{\mathcal{S}} \sum_{n=1}^{N} \varepsilon_n' + \sum_{n=1}^{N} \|\boldsymbol{t}(c_n)\|_2^2 \\
&\text{subject to} && (\boldsymbol{t}(c_1) - \boldsymbol{t}(c_2))^\top \boldsymbol{f}_{\boldsymbol{\alpha}}(\boldsymbol{x}_n) \geq p - \varepsilon_{n,r} \\
& && \quad \text{for all } n = 1, \dots, N \text{ and } (c_1, c_2) \in \mathcal{R}_n \\
& && (\boldsymbol{t}(c_1) - \boldsymbol{t}(c_2))^\top \boldsymbol{f}_{\boldsymbol{\alpha}}(\boldsymbol{x}_n) \geq 1 - \varepsilon_n' \\
& && \quad \text{for all } n = 1, \dots, N \text{ and } (c_1, c_2) \in \mathcal{S}_n
\end{aligned}
\tag{9}
$$

where $\boldsymbol{f}_{\boldsymbol{\alpha}}(\boldsymbol{x}) := \sum_{j=1}^{J} \alpha_j h_j(\boldsymbol{x})$ and $H = \{h_j : X \to \mathbb{R}^L \mid j = 1, \dots, J\}$.

In this formulation one tries to find a code $\boldsymbol{t}$ and an embedding $\boldsymbol{f}$, such that for each example the output wrt. each class this example has a relation with, reflects the order of this relations (i.e. the examples get ranked appropriately). Furthermore, the program tries to achieve a "large margin" between relevant and irrelevant classes for each sample. Similar formulations can be found in [8] (see also [11]).

Optimization of (9) is analogous to the column generation approach discussed in Section 3. We omit details due to constraints on space. A small toy example, again as a limited proof of concept, is given in Figure 2.

**Connection to Ranking Techniques** Ordinal regression through large margins [10] can be seen as an extreme case of (9), where we have as many classes as observations, and each pair of observations has to satisfy a ranking relation $f(\boldsymbol{x}_i) - f(\boldsymbol{x}_j) \geq \rho - \epsilon_{ij}$, if $\boldsymbol{x}_i$ is to be preferred to $\boldsymbol{x}_j$. This formulation can of course also be understood as a special case of multi-dimensional regression.

## 5   Conclusion

We proposed an algorithm to simultaneously optimize output codes and the embedding of the sample into the code book space building upon the notion of large margins. Further-

more, we have shown, that only quadratic and related distance measures in the code book space will lead to convex constraints and hence convex optimization problems whenever either the code or the embedding is held fixed. This is desirable since at least for these sub-problems there exist fairly efficient techniques to compute these (of course the combined optimization problem of finding the code and the embedding is not convex and has local minima). We proposed a column generation technique for solving the embedding optimization problems. It allows the use of a two-class algorithm, of which there exists many efficient ones, and has connection to boosting. Finally we proposed a technique along the same lines that should be favorable when dealing with many classes or even empty classes. Future work will concentrate on finding more efficient algorithms to solve the optimization problem and on more carefully evaluating their performance.

**Acknowledgements**   We thank B. Williamson and A. Torda for interesting discussions.

## Footnotes

[1]We could also use ternary codes, i.e. $\{-1, 0, 1\}$, allowing for "don't care" classes.

[2]However, also note that this is quite a bit worse than the best packing, which scales with $C^{-\frac{1}{L-1}}$ rather than $C^{-\frac{2}{L-1}}$. This is due a the union-bound argument in the proof, which requires us to sum over the probability that all $C(C-1)/2$ *pairs* have more than $\epsilon$ distance.

## References

[1] E.L. Allwein, R.E. Schapire, and Y. Singer. Reducing multiclass to binary: A unifying approach for margin classifiers. *Journal of Machine Learning Research*, 1:113–141, 2000.

[2] K.P. Bennett, A. Demiriz, and J. Shawe-Taylor. A column generation algorithm for boosting. In P. Langley, editor, *Proc. 17th ICML*, pages 65–72, San Francisco, 2000. Morgan Kaufmann.

[3] B. Caputo and G. Rätsch. Adaptive codes for visual categories. November 2002. Unpublished manuscript. Partial results presented at NIPS'02.

[4] K. Crammer and Y. Singer. On the learnability and design of output codes for multiclass problems. In N. Cesa-Bianchi and S. Goldberg, editors, *Proc. Colt*, pages 35–46, San Francisco, 2000. Morgan Kaufmann.

[5] O. Dekel and Y. Singer. Multiclass learning by probabilistic embeddings. In *NIPS*, vol. 15. MIT Press, 2003.

[6] T.G. Dietterich and G. Bakiri. Solving multiclass learning problems via error-correcting output codes. *Journal of Aritifical Intelligence Research*, 2:263–286, 1995.

[7] V. Guruswami and A. Sahai. Multiclass learning, boosing, and error-correcting codes. In *Proc. of the twelfth annual conference on Computational learning theory*, pages 145–155, New York, USA, 1999. ACM Press.

[8] S. Har-Peled, D. Roth, and D. Zimak. Constraint classification: A new approach to multiclass classification and ranking. In *NIPS*, vol. 15. MIT Press, 2003.

[9] T.J. Hastie and R.J. Tibshirani. Classification by pairwise coupling. In M.I. Jordan, M.J. Kearnsa, and S.A. Solla, editors, *Advances in Neural Information Processing Systems*, vol. 10. MIT Press, 1998.

[10] R. Herbrich, T. Graepel, and K. Obermayer. Large margin rank boundaries for ordinal regression. In A. J. Smola, P. L. Bartlett, B. Schölkopf, and D. Schuurmans, editors, *Advances in Large Margin Classifiers*, pages 115–132, Cambridge, MA, 2000. MIT Press.

[11] R. Jin and Z. Ghahramani. Learning with multiple labels. In *NIPS*, vol. 15. MIT Press, 2003.

[12] S. Nash and A. Sofer. *Linear and Nonlinear Programming*. McGraw-Hill, New York, NY, 1996.

[13] G. Rätsch, A. Demiriz, and K. Bennett. Sparse regression ensembles in infinite and finite hypothesis spaces. *Machine Learning*, 48(1-3):193–221, 2002. Special Issue on New Methods for Model Selection and Model Combination.

[14] G. Rätsch, M. Warmuth, S. Mika, T. Onoda, S. Lemm, and K.-R. Müller. Barrier boosting. In *Proc. COLT*, pages 170–179, San Francisco, 2000. Morgan Kaufmann.

[15] R.E. Schapire. Using output codes to boost multiclass learning problems. In *Machine Learning: Proceedings of the 14th International Conference*, pages 313–321, 1997.

[16] B. Schölkopf, A. Smola, R.C. Williamson, and P.L. Bartlett. New support vector algorithms. *Neural Computation*, 12:1207 – 1245, 2000.

[17] N. Sloane. Personal homepage. http://www.research.att.com/~njas/.

[18] W. Utschick. *Error-Correcting Classification Based on Neural Networks*. Shaker, 1998.

[19] W. Utschick and W. Weichselberger. Stochastic organization of output codes in multiclass learning problems. *Neural Computation*, 13(5):1065–1102, 2001.

[20] V.N. Vapnik and A.Y. Chervonenkis. A note on one class of perceptrons. *Automation and Remote Control*, 25, 1964.

[21] J. Weston and C. Watkins. Multi-class support vector machines. Technical Report CSD-TR-98-04, Royal Holloway, University of London, Egham, 1998.
